# A Three Tiered Approach for Articulated Object Action Modeling and Recognition

**Le Lu    Gregory D. Hager**
Department of Computer Science
Johns Hopkins University
Baltimore, MD 21218
lelu/hager@cs.jhu.edu

**Laurent Younes**
Center of Imaging Science
Johns Hopkins University
Baltimore, MD 21218
younes@cis.jhu.edu

## Abstract

Visual action recognition is an important problem in computer vision. In this paper, we propose a new method to probabilistically model and recognize actions of articulated objects, such as hand or body gestures, in image sequences. Our method consists of three levels of representation. At the low level, we first extract a feature vector invariant to scale and in-plane rotation by using the Fourier transform of a circular spatial histogram. Then, spectral partitioning [20] is utilized to obtain an initial clustering; this clustering is then refined using a temporal smoothness constraint. Gaussian mixture model (GMM) based clustering and density estimation in the subspace of linear discriminant analysis (LDA) are then applied to thousands of image feature vectors to obtain an intermediate level representation. Finally, at the high level we build a temporal multi-resolution histogram model for each action by aggregating the clustering weights of sampled images belonging to that action. We discuss how this high level representation can be extended to achieve temporal scaling invariance and to include Bi-gram or Multi-gram transition information. Both image clustering and action recognition/segmentation results are given to show the validity of our three tiered representation.

## 1   Introduction

Articulated object action modeling, tracking and recognition has been an important research issue in computer vision community for decades. Past approaches [3, 13, 4, 6, 23, 2] have used many different kinds of direct image observations, including color, edges, contour or moments [14], to fit a hand or body's shape model and motion parameters.

In this paper, we propose to learn a small set of object appearance descriptors, and then to build an aggregated temporal representation of clustered object descriptors over time. There are several obvious reasons to base gesture or motion recognition on a time sequence of observations. First, most hand or body postures are ambiguous. For example, in American Sign Language, 'D' and 'G', 'H' and 'U' have indistinguishable appearance from some viewpoints. Furthermore, these gestures are difficult to track from frame to frame due to motion blur, lack of features, and complex self-occlusions. By modeling hand/body gesture as a sequential learning problem, appropriate discriminative information can be retrieved and more action categories can be handled.

In related work, Darrell and Pentland [7] describe dynamic time warping (DTW) to align and recognize a space-time gesture against a stored library. To build the library, key views are selected from incoming an video by choosing views that have low correlation with all current views. This approach is empirical and does not guarantee any sort of global consistency of the chosen views. As a result, recognition may be unstable. In comparision, our method describes image appearances uniformly and clusters them globally from a training set containing different gestures.

For static hand posture recognition, Tomasi et al. [24] apply vector quantization methods to cluster images of different postures and different viewpoints. This is a feature-based approach, with thousands of features extracted for each image. However, clustering in a high dimensional space is very difficult and can be unstable. We argue that fewer, more global features are adequate for the purposes of gesture recognition. Furthermore, the circular histogram representation has adjustable spatial resolution to accomodate differing appearance complexities, and it is translation, rotation, and scale invariant.

In other work, [27, 9] recognize human actions at a distance by computing motion information between images and relying on temporal correlation on motion vectors across sequences. Our work also makes use of motion information, but does not rely exclusively on it. Rather, we combine appearance and motion cues to increase sensitivity beyond what either can provide alone. Since our method is based on the temporal aggregation of image clusters as a histogram to recognize an action, it can also be considered to be a temporal texton-like method [17, 16]. One advantage of the aggregated histogram model in a time-series is that it is straightforward to accommodate temporal scaling by using a sliding window. In addition, higher order models corresponding to bigrams or trigrams of simpler "gestemes" can also be naturally employed to extend the descriptive power of the method.

In summary, there are four principal contributions in this paper. First, we propose a new scale/rotation-invariant hand image descriptor which is stable, compact and representative. Second, we introduce a methods for sequential smoothing of clustering results. Third, we show LDA/GMM with spectral partitioning initialization is an effective way to learn well-formed probability densities for clusters. Finally, we recognize image sequences as actions efficiently based on a flexible histogram model. We also discuss improvement to the method by incorporating motion information.

## 2   A Three Tiered Approach

We propose a three tiered approach for dynamic action modeling comprising low level feature extraction, intermediate level feature vector clustering and high level histogram recognition as shown in Figure 1.

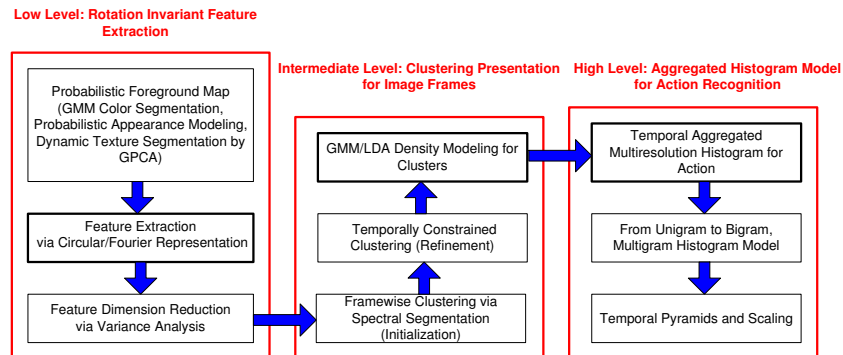

Figure 1: Diagram of a three tier approach for dynamic articulated object action modeling.

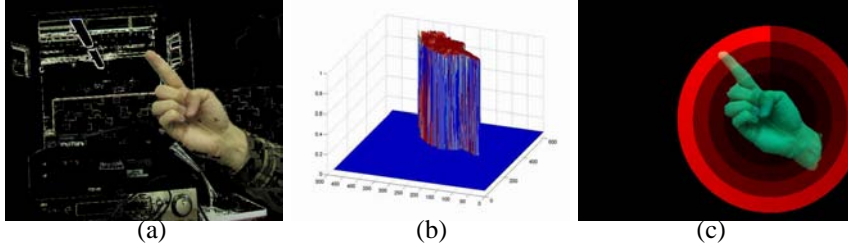

|  (a)  |  (b)  |  (c)  |

Figure 2: (a) Image after background subtraction (b) GMM based color segmentation (c) Circular histogram for feature extraction.

## 2.1 Low Level: Rotation Invariant Feature Extraction

In the low level image processing, our goals are to locate the region of interest in an image and to extract a scale and in-plane rotation invariant feature vector as its descriptor. In order to accomplish this, a reliable and stable foreground model of the target in question is expected. Depending on the circumstances, a Gaussian mixture model (GMM) for segmentation [15], probabilistic appearance modeling [5], or dynamic object segmentation by Generalized Principal Component Analysis (GPCA) [25] are possible solutions. In this paper, we apply a GMM for hand skin color segmentation.

We fit a GMM by first performing a simple background subtraction to obtain a noisy foreground containing a hand object (shown in Figure 2 (a)). From this, more than 1 million RGB pixels are used to train skin and non-skin color density models with 10 Gaussian kernels for each class. Having done this, for new images a probability density ratio $P_{skin}/P_{nonskin}$ of these two classes is computed. If $P_{skin}/P_{nonskin}$ is larger than 1, the pixel is considered as skin (foreground) and is otherwise background. A morphological operator is then used to clean up this initial segmentaion and create a binary mask for the hand object. We then compute the centroid and second central moments of this 2D mask. A circle is defined about the target by setting its center as the centroid and its radius as 2.8 times largest eigenvalues of the second central moment matrix (covering over 99% skin pixels in Figure 2 (c)). This circle is then divided to have 6 concentric annuli which contain 1, 2, 4, 8, 16, 32 bins from inner to outer, respectively. Since the position and size of this circular histogram is determined by the color segmentation, it is translation and scale invariant.

We then normalize the density value $P_{skin} + P_{nonskin} = 1$ for every pixel within the foreground mask (Figure 2) over the hand region. For each bin of the circular histogram, we calculate the mean of $P_{skin}$ ( $-log(P_{skin})$, or $-log(P_{skin}/P_{nonskin})$ are also possible choices) of pixels in that bin as its value. The values of all bins along each circle form a vector, and 1D Fourier transform is applied to this vector. The power spectra of all annuli are ordered into a linear list producing a feature vector $\vec{f}(t)$ of 63 dimensions representing the appearance of a hand image.[1] Note that the use of the Fourier power spectrum of the annuli makes the representation rotation invariant.

## 2.2 Intermediate Level: Clustering Presentation for Image Frames

After the low level processing, we obtain a scale and rotation invariant feature vector as an appearance representation for each image frame. The temporal evolution of feature vectors represent actions. However, not all the images are actually unique in appearance.

At the intermediate level, we cluster images from a set of feature vectors. This frame-wise clustering is critical for dimension reduction and the stability of high level recognition.

**Initializing Clusters by Spectral Segmentation**   There are two critical problems with clustering algorithms: determining the true number of clusters and initializing each cluster. Here we use a spectral clustering method [20, 22, 26, 18] to solve both problems. We first build the affinity matrix of pairwise distances between feature fectors[2]. We then perform a singular value decomposition on the affinity matrix with proper normalization [20]. The number of clusters is determined by choosing the $n$ dominant eigenvalues. The corresponding eigenvectors are taken as an orthogonal subspace for all the data.

To get $n$ cluster centers, we take the approach of [20] and choose vectors that minimize the absolute value of cosine between any two cluster centers:

$$ID(k) = \begin{cases} \text{rand}(0, N) & : & k = 1 \\ \arg\min_{t=1..N} \sum_{c=1}^{k-1} |\cos(\vec{f^n}(ID(c)), \vec{f^n}(t))| & : & n \geq k > 1 \end{cases} \quad (1)$$

where $\vec{f^n}(t)$ is the feature vector of image frame $t$ after numerical normalization in [20] and $ID(k)$ is the image frame number chosen for the center of cluster $k$. $N$ is the number of images used for spectral clustering. For better clustering results, multiple restarts are used for initialization.

Unlike [18], we find this simple clustering procedure is sufficient to obtain a good set of clusters from only a few restarts. After initialization, the Kmeans [8] is used to smooth the centers. Let $C_1(t)$ denote the class label for image $t$, and $\vec{g}(c) = \vec{f}(ID(c));\ c = 1 \ldots n$ denote cluster centers.

**Refinement: Temporally Constrained Clustering**   Spectral clustering methods are designed for an unordered "bag" of feature vectors, but, in our case, the temporal ordering of image is an important source of information. In particular, the stablity of appearance is easily computed by computing the motion energy[3] between two frames. Let $M(t)$ denote the motion energy between frames $t$ and $t-1$. Define $T_{k,j} = \{t | C_1(t) = k, C_1(t-1) = j\}$ and $\bar{M}(k,j) = \sum_{t \in T_{k,j}} M(t)/|T_{k,j}|$. We now create a regularized clustering cost function as

$$C_2(t) = \arg\max_{c=1..n} \left\{ \frac{e^{-\|f(t)-g(c)\|}}{\sum_{c=1}^{n} e^{-\|f(t)-g(c)\|}} + \lambda \frac{e^{-\frac{\|g(c)-g(C_2(t-1))\|}{M(t)}}}{\sum_{c=1}^{n} e^{-\frac{\|g(c)-g(C_2(t-1))\|}{M(c, C_2(t-1))}}} \right\} \quad (2)$$

where $\lambda$ is the weighting parameter. Here motion energy $M(t)$ plays a role as the temperature $T$ in simulated annealing. When it is high (strong motion between frames), the motion continuity condition is violated and the labels of successive frames can change freely; when it is low, the smoothness term constrains the possible transitions of classes with low $\bar{M}(k,j)$.

With this in place, we now scan through the sequence searching for $C_2(t)$ of maximum value given $C_2(t-1)$ is already fixed. [4] This temporal smoothing is most relevant with images with motions, and static frames are already stably clustered and therefore their cluster labels to not change.

**GMM for Density Modeling and Smoothing**   Given clusters, we build a probability density model for each. A Gaussian Mixture Model [11, 8] is used to gain good local relaxation based on the initial clustering result provided by the above method and good generalization for new data. Due to the curse of dimensionality, it is difficult to obtain a good estimate of a high dimensional density function with limited and largely varied training data. We introduce an iterative method incorporating Linear Discriminative Analysis (LDA) [8] and a GMM in an EM-like fashion to perform dimensional reduction. The initial clustering labels help to build the scatter matrices for LDA. The optimal projection matrix of LDA is then obtained from the decomposition of clusters' scatter matrices [8]. The original feature vectors can be further projected into a low dimensional space, which improves the estimation of multi-variate Gaussian density function. With the new clustering result from GMM, LDA's scatter matrices and projection matrix can be re-estimated, and GMM can also be re-modeled in the new LDA subspace. This loop converges within $3 \sim 5$ iterations from our experiments. Intuitively, LDA projects the data into a low dimensional subspace where the image clusters are well separated, which helps to have a good parameter estimation for GMM with limited data. Given more accurate GMM, more accurate clustering results are obtained, which also causes better estimate of LDA. The theoretical proof of convergence is undertaken. After this process, we have a Gaussian density model for each cluster.

## 2.3   High Level: Aggregated Histogram Model for Action Recognition

Given a set of $n$ clusters, define $w(t) = [p_{c_1}(f(t)), p_{c_2}(f(t)), ..., p_{c_n}(f(t))]^T$ where $p_x(y)$ denotes the density value of the vector $y$ with respect to the GMM for cluster $x$. An action is then a trajectory of $[w(t_1), w(t_1 + 1), ..., w(t_2)]^T$ in $\Re^n$. For recognition purposes, we want to calculate some discriminative statistics from each trajectory. One natural way is to use its mean $H_{t_1,t_2} = \sum_{t=t_1}^{t_2} w(t)/(t_2 - t_1 + 1)$ over time which is a temporal weighted histogram. Note that the histogram $H_{t_1,t_2}$ bins are precisely corresponding to the trained clusters.

From the training set, we aggregate the cluster weights of images within a given hand action to form a histogram model. In this way, a temporal image sequence corresponding to one action is represented by a single vector. The matching of different actions is equivalent to compute the similarity of two histograms which has variants. Here we use Bhattacharyya similarity metric [1] which has has several useful properties including: it is an approximation of $\chi^2$ test statistics with fixed bias; it is self-consistent; it does not have the singularity problem while matching empty histogram bins; and its value is properly bounded within $[0, 1]$. Assume we have a library of action histograms $H_1^*, H_2^*, ..., H_M^*$, the class label of a new action $\hat{H}_{t_1,t_2}$ is determined by the following equation.

$$L(\hat{H}_{t_1,t_2}) = \arg \min_{l=1..M} \left\{ D(H_l^*, \hat{H}_{t_1,t_2}) = \left[ 1 - \sum_{c=1}^{n} \sqrt{H_l^*(c) * \hat{H}_{t_1,t_2}(c)} \right]^{\frac{1}{2}} \right\} \quad (3)$$

This method is low cost because only one exemplar per action category is needed.

One problem with this method is that all sequence information has been compressed, e.g., we cannot distinguish an opening hand gesture from a closing hand using only one histogram. This problem can be easily solved by subdividing the sequence and histogram model into m parts: $H_{t_1,t_2}^m = [H_{t_1,(t_1+t_2)/m}, ..., H_{(t_1+t_2)*(m-1)/m,t_2}]^T$. For an extreme case when one frame is a subsequence, the histogram model simply becomes exactly the vector form of the representative surface.

We intend to classify hand actions with speed differences into the same category. To achieve this, the image frames within a hand action can be sub-sampled to build a set of temporal pyramids. In order to segment hand gestures from a long video sequence, we create several sliding windows with different frame sampling rates. The proper time scaling magnitude is found by searching for the best fit over temporal pyramids.

Taken together, the histogram representation achieves an adjustable multi-resolution measurement to describe actions. A Hidden Markov Model (HMM) with discrete observations could be also employed to train models for different hand actions, but more template samples per gesture class are required. The histogram recognition method has the additional advantage that it does not depend on extremely accurate frame-wise clustering. A small proportion of incorrect labels does not effect the matching value much. In comparison, in an HMM with few training samples, outliers seriously impact the accuracy of learning. From the viewpoint of considering hand actions as a language process, our model is an integration of individual observations (by labelling each frame with a set of learned clusters) from different time slots. The labels' transitions between successive frames are not used to describe the temporal sequence. By subdividing the histogram, we are extending the representation to contain bigram, trigram, etc. information.

## 3  Results

We have tested our three tiered method on the problem of recognizing sequences of hand spelling gestures.

**Framewise clustering.** We first evaluate the low level representation of single images and intermediate clustering algorithms. A training set of $3015$ images are used. The frame-to-frame motion energy is used to label images as static or dynamic. For spectral clustering, $3 \sim 4$ restarts from both the dynamic and static set are sufficient to cover all the modes in the training set. Then, temporal smoothing is employed and a Gaussian density is calculated for each cluster in a $10$ dimensional subspace of the LDA projection. As a result, $24$ clusters are obtained which contain $16$ static and $8$ dynamic modes. Figure 3 shows $5$ frames closest to the mean of the probability density of cluster $1, 3, 19, 5, 13, 8, 21, 15, 6, 12$. It can be seen that clustering results are insensitive to artifacts of skin segmentation. From Figure 3, it is also clear that dynamic modes have significantly larger determinants than static ones. The study of the eigenvalues of covariance matrices shows that their super-ellipsoid shapes are expanded within $2 \sim 3$ dimensions or $6 \sim 8$ dimensions for static or dynamic clusters. Taken together, this means that static clusters are quite tight, while dynamic clusters contain much more in-class variation. From Figure 4 (c), dynamic clusters gain more weight during the smoothing process incorporating the temporal constraint and subsequent GMM refinement.

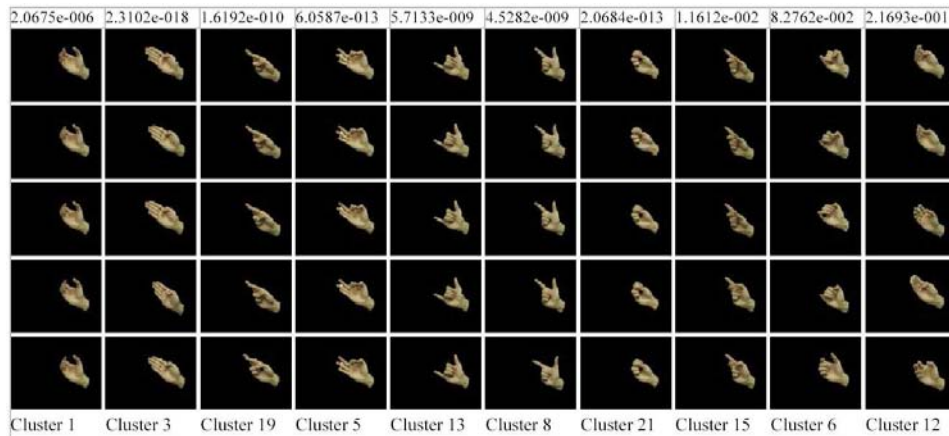

Figure 3: Image clustering results after low and intermediate level processing.

**Action recognition and segmentation.** For testing images, we first project their feature

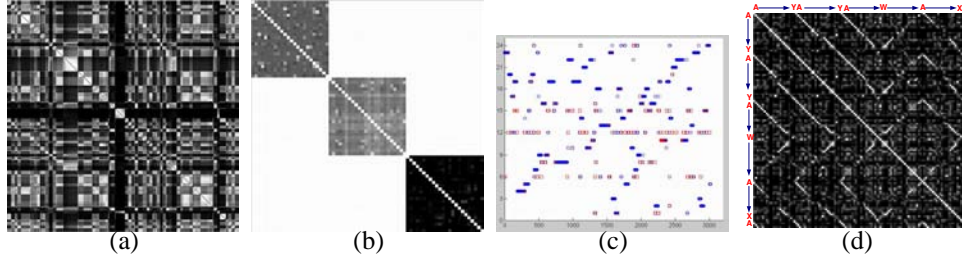

Figure 4: (a) Affinity matrix of 3015 images. (b) Affinity matrices of cluster centoids (from upper left to lower right) after spectral clustering, temporal smoothing and GMM. (c) Labelling results of 3015 images (red squares are frames whose labels changed with smoothing process after spectral clustering). (d) The similarity matrix of segmented hand gestures. The letters are labels of gestures.

vectors into the LDA subspace. Then, the GMM is used to compute their weights with respect to each cluster. We manually choose 100 sequences for testing purposes, and compute their similarities with respect to a library of 25 gestures. The length of the action sequences was $9 \sim 38$ frames. The temporal scale of actions in the same category ranged from 1 to 2.4. The results were recognition rates of $90\%$ and $93\%$ without/with temporal smoothing (Equation 2). Including the top three candidates, the recognition rates increase to $94\%$ and $96\%$, respectively. We also used the learned model and a sliding window with temporal scaling to segment actions from a 6034 frame video sequence containing dynamic gestures and static hand postures. The similarity matrices among 123 actions found in the video is shown in Figure 4 (d). 106 out of 123 actions ($86.2\%$) are correctly segmented and recognized.

**Integrating motion information.** As noted previously, our method cannot distinguish opening/closing hand gestures without temporally subdividing histograms. An alternative solution is to integrate motion information[5] between frames. Motion feature vectors are also clustered, which results a joint (appearance and motion) histogram model for actions. We assume independence of the data and therefore simple contatenate these two histograms into a single action representation. From our preliminary experiments, both motion integration and histogram subdivision are comparably effective to recognize gestures with opposite direction.

## 4   Conclusion and Discussion

We have presented a method for classifying the motion of articulated gestures using LDA/GMM-based clustering methods and a histogram-based model of temporal evolution. Using this model, we have obtained extremely good recognition results using a relatively coarse representation of appearance and motion in images.

There are mainly three methods to improve the performance of histogram-based classification, i.e., adaptive binning, adaptive subregion, and adaptive weighting [21]. In our approach, adaptive binning of the histogram is automatically learned by our clustering algorithms; adaptive subregion is realized by subdividing action sequences to enrich the histogram's descriptive capacity in the temporal domain; adaptive weighting is achieved from the trained weights of Gaussian kernels in GMM.

Our future work will focus on building a larger hand action database containing $50 \sim 100$

categories for more extensive testing, and on extending the representation to include other types of image information (e.g. contour information). Also, by finding an effective foreground segmentation module, we intend to apply the same methods to other applications such as recognizing stylized human body motion.

## Footnotes

[1]An optional dimension reduction of feature vectors can be achieved by eliminating dimensions which have low variance. It means that feature values of those dimensions do not change much in the data, therefore are non-informative.

[2]The exponent of either Euclidean distance or Cosine distance between two feature vectors can be used in this case.

[3]A simple method is to compute motion energy as the Sum of Squared Differences (SSD) by subtracting two $P_{skin}$ density masses from successive images.

[4]Note that $\bar{M}(k,j)$ changes after scanning the labels of the image sequence once, thus more iterations could be used to achieve more accurate temporal smoothness of $C_3(t), t = 1..N$. From our experiments, more iterations does not change the result much.

[5]Motion information can be extracted by first aligning two hand blobs, subtracting two skin-color density masses, then using the same circular histogram in section 2.1 to extract a feature vector for positive and negative density residues respectively. Another simple way is to subtract two frames' feature vectors directly.

## References

[1] F. Aherne, N. Thacker, and P. Rockett, The Bhattacharyya Metric as an Absolute Similarity Measure for Frequency Coded Data, *Kybernetika*, **34:4**, pp. 363-68, 1998.

[2] V. Athitsos and S. Sclaroff, Estimating 3D Hand Pose From a Cluttered Image, *CVPR*, 2003.

[3] M. Brand, Shadow Puppetry, *ICCV*, 1999.

[4] R. Bowden and M. Sarhadi, A Non-linear of Shape and Motion for Tracking Finger Spelt American Sign Language, *Image and Vision Computing*, **20:597-607**, 2002.

[5] T. Cootes, G. Edwards and C. Taylor, Active Appearance Models, *IEEE Trans. PAMI*, **23:6**, pp. 681-685, 2001.

[6] D. Cremers, T. Kohlberger and C. Schnrr, Shape statistics in Kernel Space for Variational Image Segmentation, *Pattern Recognition*, **36:1929-1943**, 2003.

[7] T. J. Darrell and A. P. Pentland, Recognition of Space-Time Gestures using a Distributed Representation, MIT Media Laboratory Vision and Modeling TR-197.

[8] R. O. Duda, P. E. Hart and D. G. Stork, *Pattern Classification*, Wiley Interscience, 2002.

[9] A. Efros, A. Berg, G. Mori and J. Malik, Recognizing Action at a Distance. *ICCV*, pp. 726–733, 2003.

[10] W. T. Freeman and E. H. Adelson, The Design and Use of Steerable Filters, IEEE Trans. PAMI, **13:9**, pp. 891-906, 1991.

[11] T. Hastie and R. Tibshirani, Discriminant Analysis by Gaussian Mixtures. *Journal of Royal Statistical Society Series B*, 58(1):155-176.

[12] W. Hawkins, P. Leichner and N. Yang, The Circular Harmonic Transform for SPECT Reconstruction and Boundary Conditions on the Fourier Transform of the Sinogram, *IEEE Trans. on Medical Imaging*, **7:2**, 1988.

[13] A. Heap and D. Hogg, Wormholes in Shape Space: Tracking through Discontinuous Changes in Shape, *ICCV*, 1998.

[14] M. K. Hu, Visual pattern recognition by moment invariants, *IEEE Trans. Inform. Theory*, **8:179-187**, 1962.

[15] M. J. Jones and J. M. Rehg, Statistical Color Models with Application to Skin Detection *Int. J. of Computer Vision*, **46:1** pp: 81-96, 2002.

[16] B. Julesz, Textons, the elements of texture perception, and their interactions. *Nature*, 290:91-97, 1981.

[17] T. Leung and J. Malik, Representing and Recognizing the Visual Appearance of Materials using Three-dimensional Textons, *Int. Journal of Computer Vision*, **41:1**, pp. 29-44, 2001.

[18] M. Maila and J. Shi, Learning Segmentation with Random Walk, *NIPS* 2001.

[19] B. Moghaddam and A. Pentland, Probabilistic Visual Learning for Object Representation, *IEEE Trans. PAMI* **19:7**, 1997.

[20] A. Ng, M. Jordan and Y. Weiss, On Spectral Clustering: Analysis and an algorithm, *NIPS*, 2001.

[21] S. Satoh, Generalized Histogram: Empirical Optimization of Low Dimensional Features for Image Matching, *ECCV*, 2004.

[22] J. Shi and J. Malik, Normalized Cuts and Image Segmentation, *IEEE Trans. on PAMI*, 2000.

[23] B. Stenger, A. Thayananthan, P. H. S. Torr, and R. Cipolla, Filtering Using a Tree-Based Estimator, *ICCV*, **II:1063-1070**, 2003.

[24] C. Tomasi, S. Petrov and A. Sastry, 3D tracking = classification + interpolation, *ICCV*, 2003.

[25] R. Vidal and R. Hartley, Motion Segmentation with Missing Data using PowerFactorization and GPCA, *CVPR*, 2004.

[26] Y. Weiss, Segmentation using eigenvectors: A Unifying view. *ICCV*, 1999.

[27] Lihi Zelnik-Manor and Michal Irani, Event-based video analysis, *CVPR*, 2001.
